# Training Data Selection
# for Optimal Generalization
# in Trigonometric Polynomial Networks

Masashi Sugiyama*and Hidemitsu Ogawa
Department of Computer Science, Tokyo Institute of Technology,
2-12-1, O-okayama, Meguro-ku, Tokyo, 152-8552, Japan.
*sugi@cs.titech.ac.jp*

## Abstract

In this paper, we consider the problem of active learning in trigonometric polynomial networks and give a necessary and sufficient condition of sample points to provide the optimal generalization capability. By analyzing the condition from the functional analytic point of view, we clarify the mechanism of achieving the optimal generalization capability. We also show that a set of training examples satisfying the condition does not only provide the optimal generalization but also reduces the computational complexity and memory required for the calculation of learning results. Finally, examples of sample points satisfying the condition are given and computer simulations are performed to demonstrate the effectiveness of the proposed active learning method.

## 1 Introduction

*Supervised learning* is obtaining an underlying rule from training examples, and can be formulated as a function approximation problem. If sample points are actively designed, then learning can be performed more efficiently. In this paper, we discuss the problem of designing sample points, referred to as *active learning*, for optimal generalization.

Active learning is classified into two categories depending on the optimality. One is *global optimal*, where a set of all training examples is optimal (e.g. Fedorov [3]). The other is *greedy optimal*, where the next training example to sample is optimal in each step (e.g. MacKay [5], Cohn [2], Fukumizu [4], and Sugiyama and Ogawa [10]). In this paper, we focus on the global optimal case and give a new active learning method in trigonometric polynomial networks. The proposed method does not employ any approximations in its derivation, so that it provides exactly the optimal generalization capability. Moreover, the proposed method reduces the computational complexity and memory required for the calculation of learning results. Finally, the effectiveness of the proposed method is demonstrated through computer simulations.

# 2 Formulation of supervised learning

In this section, the supervised learning problem is formulated from the functional analytic point of view (see Ogawa [7]). Then, our learning criterion and model are described.

## 2.1 Supervised learning as an inverse problem

Let us consider the problem of obtaining the optimal approximation to a target function $f(x)$ of $L$ variables from a set of $M$ training examples. The training examples are made up of sample points $x_m \in \mathcal{D}$, where $\mathcal{D}$ is a subset of the $L$-dimensional Euclidean space $\mathbf{R}^L$, and corresponding sample values $y_m \in \mathbf{C}$:

$$\{(x_m, y_m) \mid y_m = f(x_m) + n_m\}_{m=1}^M, \tag{1}$$

where $y_m$ is degraded by zero-mean additive noise $n_m$. Let $n$ and $y$ be $M$-dimensional vectors whose $m$-th elements are $n_m$ and $y_m$, respectively. $y$ is called a *sample value vector*. In this paper, the target function $f(x)$ is assumed to belong to a *reproducing kernel Hilbert space* $H$ (Aronszajn [1]). If $H$ is unknown, then it can be estimated by model selection methods (e.g. Sugiyama and Ogawa [9]). Let $K(\cdot, \cdot)$ be the reproducing kernel of $H$. If a function $\psi_m(x)$ is defined as $\psi_m(x) = K(x, x_m)$, then the value of $f$ at a sample point $x_m$ is expressed as $f(x_m) = \langle f, \psi_m \rangle$, where $\langle \cdot, \cdot \rangle$ stands for the inner product. For this reason, $\psi_m$ is called a *sampling function*. Let $A$ be an operator defined as

$$A = \sum_{m=1}^M \left( e_m \otimes \overline{\psi_m} \right), \tag{2}$$

where $e_m$ is the $m$-th vector of the so-called standard basis in $\mathbf{C}^M$ and $(\cdot \otimes \overline{\cdot})$ stands for the *Neumann-Schatten product*[1]. $A$ is called a *sampling operator*. Then, the relationship between $f$ and $y$ can be expressed as

$$y = Af + n. \tag{3}$$

Let us denote a mapping from $y$ to a learning result $f_0$ by $X$:

$$f_0 = Xy, \tag{4}$$

where $X$ is called a *learning operator*. Then, the supervised learning problem is reformulated as an inverse problem of obtaining $X$ providing the best approximation $f_0$ to $f$ under a certain learning criterion.

## 2.2 Learning criterion and model

As mentioned above, function approximation is performed on the basis of a learning criterion. Our purpose of learning is to minimize the *generalization error* of the learning result $f_0$ measured by

$$J_G = E_n \|f_0 - f\|^2, \tag{5}$$

where $E_n$ denotes the ensemble average over noise. In this paper, we adopt *projection learning* as our learning criterion. Let $A^*$, $\mathcal{R}(A^*)$, and $P_{\mathcal{R}(A^*)}$ be the adjoint operator of $A$, the range of $A^*$, and the orthogonal projection operator onto $\mathcal{R}(A^*)$, respectively. Then, projection learning is defined as follows.

**Definition 1 (Projection learning)** *(Ogawa [6]) An operator $X$ is called the projection learning operator if $X$ minimizes the functional $J_P[X] = E_n\|Xn\|^2$ under the constraint $XA = P_{\mathcal{R}(A^*)}$.*

It is well-known that Eq.(5) can be decomposed into the *bias* and *variance*:

$$J_G = \|P_{\mathcal{R}(A^*)}f - f\|^2 + E_n\|Xn\|^2. \qquad (6)$$

Eq.(6) implies that the projection learning criterion reduces the bias to a certain level and minimizes the variance.

Let us consider the following function space.

**Definition 2 (Trigonometric polynomial space)** *Let $x = (\xi^{(1)}, \xi^{(2)}, \cdots, \xi^{(L)})^\top$. For $1 \le l \le L$, let $N_l$ be a positive integer and $\mathcal{D}_l = [-\pi, \pi]$. Then, a function space $H$ is called a trigonometric polynomial space of order $(N_1, N_2, \cdots, N_L)$ if $H$ is spanned by*

$$\left\{ \prod_{l=1}^{L} \exp(in_l\xi^{(l)}) \right\}_{n_1=-N_1, n_2=-N_2, \cdots, n_L=-N_L}^{N_1, N_2, \cdots, N_L} \qquad (7)$$

*defined on $\mathcal{D}_1 \times \mathcal{D}_2 \times \cdots \times \mathcal{D}_L$, and the inner product in $H$ is defined as*

$$\langle f, g \rangle = \frac{1}{(2\pi)^L} \int_{-\pi}^{\pi} \int_{-\pi}^{\pi} \cdots \int_{-\pi}^{\pi} f(x)\overline{g(x)} d\xi^{(1)} d\xi^{(2)} \cdots d\xi^{(L)}. \qquad (8)$$

The dimension $\mu$ of a trigonometric polynomial space of order $(N_1, N_2, \cdots, N_L)$ is $\mu = \prod_{l=1}^{L}(2N_l + 1)$, and the reproducing kernel of this space is expressed as

$$K(x, x') = \prod_{l=1}^{L} K_l(\xi^{(l)}, \xi^{(l)'}), \qquad (9)$$

where

$$K_l(\xi^{(l)}, \xi^{(l)'}) = \begin{cases} \sin\frac{(2N_l+1)(\xi^{(l)}-\xi^{(l)'})}{2} \Big/ \sin\frac{\xi^{(l)}-\xi^{(l)'}}{2} & \text{if } \xi^{(l)} \ne \xi^{(l)'}, \\ 2N_l+1 & \text{if } \xi^{(l)} = \xi^{(l)'}. \end{cases} \qquad (10)$$

## 3  Active learning in trigonometric polynomial space

The problem of active learning is to find a set $\{x_m\}_{m=1}^{M}$ of sample points providing the optimal generalization capability. In this section, we give the optimal solution to the active learning problem in the trigonometric polynomial space.

Let $A^\dagger$ be the *Moore-Penrose generalized inverse*[2] of $A$. Then, the following proposition holds.

**Proposition 1** *If the noise covariance matrix $Q$ is given as $Q = \sigma^2 I$ with $\sigma^2 > 0$, then the projection learning operator $X$ is expressed as $X = A^\dagger$.*

Note that the sampling operator $A$ is uniquely determined by $\{x_m\}_{m=1}^{M}$ (see Eq.(2)).

From Eq.(6), the bias of a learning result $f_0$ becomes zero for all $f$ in $H$ if and only if $\mathcal{N}(A) = \{0\}$, where $\mathcal{N}(\cdot)$ stands for the null space of an operator. For this reason,

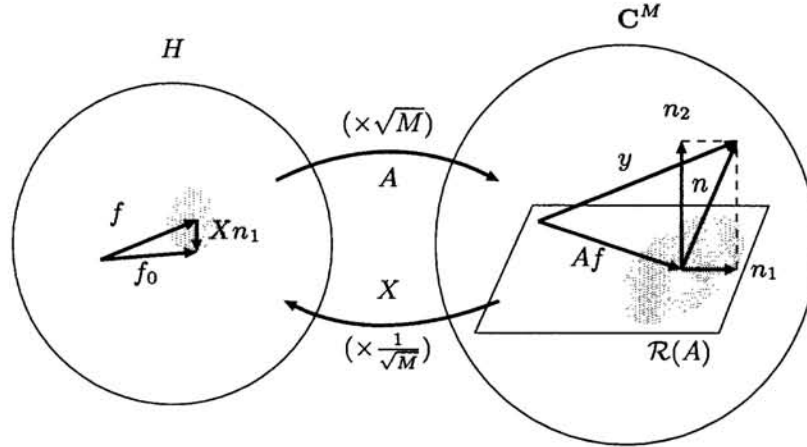

Figure 1: Mechanism of noise suppression by Theorem 1. If a set $\{x_m\}_{m=1}^M$ of sample points satisfies $A^*A = MI$, then $XAf = f$, $\|Xn_1\| = \frac{1}{\sqrt{M}}\|n_1\|$, and $Xn_2 = 0$.

we consider the case where a set $\{x_m\}_{m=1}^M$ of sample points satisfies $\mathcal{N}(A) = \{0\}$. In this case, Eq.(6) is reduced to

$$J_G = E_n\|A^\dagger n\|^2, \tag{11}$$

which is equivalent to the noise variance in $H$. Consequently, the problem of active learning becomes the problem of finding a set $\{x_m\}_{m=1}^M$ of sample points minimizing Eq.(11) under the constraint $\mathcal{N}(A) = \{0\}$.

First, we derive a condition for optimal generalization in terms of the sampling operator $A$.

**Theorem 1** *Assume that the noise covariance matrix $Q$ is given as $Q = \sigma^2 I$ with $\sigma^2 > 0$. Then, $J_G$ in Eq.(11) is minimized under the constraint $\mathcal{N}(A) = \{0\}$ if and only if*

$$A^*A = MI, \tag{12}$$

*where $I$ denotes the identity operator on $H$. In this case, the minimum value of $J_G$ is $\sigma^2\mu/M$, where $\mu$ is the dimension of $H$.*

Eq.(12) implies that $\{\frac{1}{\sqrt{M}}\psi_m\}_{m=1}^M$ forms a *pseudo orthonormal basis* (Ogawa [8]) in $H$, which is an extension of orthonormal bases. The following lemma gives interpretation of Theorem 1.

**Lemma 1** *When a set $\{x_m\}_{m=1}^M$ of sample points satisfies Eq.(12), it holds that*

$$XAf = f \quad \text{for all } f \in H, \tag{13}$$

$$\|Af\| = \sqrt{M}\|f\| \quad \text{for all } f \in H, \tag{14}$$

$$\|Xu\| = \begin{cases} \frac{1}{\sqrt{M}}\|u\| & \text{for } u \in \mathcal{R}(A), \\ 0 & \text{for } u \in \mathcal{R}(A)^\perp. \end{cases} \tag{15}$$

Eqs.(14) and (15) imply that $\frac{1}{\sqrt{M}}A$ becomes an *isometry* and $\sqrt{M}X$ becomes a *partial isometry* with the initial space $\mathcal{R}(A)$, respectively. Let us decompose the noise $n$ as $n = n_1 + n_2$, where $n_1 \in \mathcal{R}(A)$ and $n_2 \in \mathcal{R}(A)^\perp$. Then, the sample value vector $y$ is rewritten as $y = Af + n_1 + n_2$. It follows from Eq.(13) that the signal component $Af$ is transformed into the original function $f$ by $X$. From Eq.(15), $X$ suppresses the magnitude of noise $n_1$ in $\mathcal{R}(A)$ by $\frac{1}{\sqrt{M}}$ and completely removes the

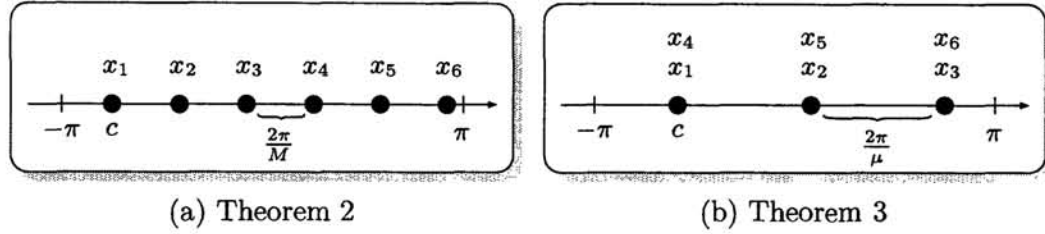

(a) Theorem 2                                    (b) Theorem 3

Figure 2: Two examples of sample points such that Condition (12) holds ($\mu = 3$ and $M = 6$).

noise $n_2$ in $\mathcal{R}(A)^\perp$. This analysis is summarized in Fig.1. Note that Theorem 1 and its interpretation are valid for all Hilbert spaces such that $K(x, x)$ is a constant for any $x$.

In Theorem 1, we have given a necessary and sufficient condition to minimize $J_G$ in terms of the sampling operator $A$. Now we give two examples of sample points $\{x_m\}_{m=1}^M$ such that Condition (12) holds. From here on, we focus on the case when the dimension $L$ of the input $x$ is 1 for simplicity. However, the following results can be easily scaled to the case when $L > 1$.

**Theorem 2** *Let $M \geq \mu$, where $\mu$ is the dimension of $H$. Let $c$ be an arbitrary constant such that $-\pi \leq c \leq -\pi + \frac{2\pi}{M}$. If a set $\{x_m\}_{m=1}^M$ of sample points is determined as*

$$x_m = c + \frac{2\pi}{M}(m - 1), \tag{16}$$

*then Eq.(12) holds.*

**Theorem 3** *Let $M = k\mu$ where $k$ is a positive integer. Let $c$ be an arbitrary constant such that $-\pi \leq c \leq -\pi + \frac{2\pi}{\mu}$. If a set $\{x_m\}_{m=1}^M$ of sample points is determined as*

$$x_m = c + \frac{2\pi}{\mu}r, \quad where \quad r = m - 1 \, (mod \, \mu), \tag{17}$$

*then Eq.(12) holds.*

Theorem 2 means that $M$ sample points are fixed to $2\pi/M$ intervals in the domain $[-\pi, \pi]$ and sample values are gathered once at each point (see Fig.2 (a)). In contrast, Theorem 3 means that $\mu$ sample points are fixed to $2\pi/\mu$ intervals in the domain and sample values are gathered $k$ times at each point (see Fig.2 (b)).

Now, we discuss calculation methods of the projection learning result $f_0(x)$. Let $h_m$ be the $m$-th column vector of the $M$-dimensional matrix $(AA^*)^\dagger$. Then, for general sample points, the projection learning result $f_0(x)$ can be calculated as

$$f_0(x) = \sum_{m=1}^M \langle y, h_m \rangle \psi_m(x). \tag{18}$$

When we use the optimal sample points satisfying Condition (12), the following theorems hold.

**Theorem 4** *When Eq.(12) holds, the projection learning result $f_0(x)$ can be calculated as*

$$f_0(x) = \frac{1}{M} \sum_{m=1}^M y_m \psi_m(x). \tag{19}$$

**Theorem 5** *When sample points are determined following Theorem 3, the projection learning result $f_0(x)$ can be calculated as*

$$f_0(x) = \frac{1}{\mu} \sum_{p=1}^{\mu} \overline{y_p} \psi_p(x), \quad where \quad \overline{y_p} = \frac{1}{k} \sum_{q=1}^{k} y_{p+\mu(q-1)}. \tag{20}$$

In Eq.(18), the coefficient of $\psi_m(x)$ is obtained by the inner product $\langle y, h_m \rangle$. In contrast, it is replaced with $y_m/M$ in Eq.(19), which implies that the Moore-Penrose generalized inverse of $AA^*$ is not required for calculating $f_0(x)$. This property is quite useful when the number $M$ of training examples is very large since the calculation of the Moore-Penrose generalized inverse of high dimensional matrices is sometimes unstable. In Eq.(20), the number of basis functions is reduced to $\mu$ and the coefficient of $\psi_p(x)$ is obtained by $\overline{y_p}/\mu$, where $\overline{y_p}$ is the mean sample values at $x_p$.

For general sample points, the computational complexity and memory required for calculating $f_0(x)$ by Eq.(18) are both $O(M^2)$. In contrast, Theorem 4 states that if a set of sample points satisfies Eq.(12), then both the computational complexity and memory are reduced to $O(M)$. Hence, Theorem 1 and Theorem 4 do not only provide the optimal generalization but also reduce the computational complexity and memory. Moreover, if we determine sample points following Theorem 3 and calculate the learning result $f_0(x)$ by Theorem 5, then the computational complexity and memory are reduced to $O(\mu)$. This is extremely efficient since $\mu$ does not depend on the number $M$ of training examples. The above results are shown in Tab.1.

## 4 Simulations

In this section, the effectiveness of the proposed active learning method is demonstrated through computer simulations.

Let $H$ be a trigonometric polynomial space of order 100, and the noise covariance matrix $Q$ be $Q = I$. Let us consider the following three sampling schemes.

**(A) Optimal sampling:** Training examples are gathered following Theorem 3.

**(B) Experimental design:** Eq.(2) in Cohn [2] is adopted as the active learning criterion. The value of this criterion is evaluated by 30 reference points. The sampling location is determined by multi-point-search with 3 candidates.

**(C) Passive learning:** Training examples are given unilaterally.

Fig.3 shows the relation between the number of training examples and the generalization error. The horizontal and vertical axes display the number of training examples and the generalization error $J_G$ measured by Eq.(5), respectively. The solid line shows the sampling scheme (A). The dashed and dotted lines denote the averages of 10 trials of the sampling schemes (B) and (C), respectively. When the number of training examples is 201, the generalization error of the sampling scheme (A) is 1 while the generalization errors of the sampling schemes (B) and (C) are $3.18 \times 10^4$ and $8.75 \times 10^4$, respectively. This graph illustrates that the proposed sampling scheme gives much better generalization capability than the sampling schemes (B) and (C) especially when the number of training examples is not so large.

## 5 Conclusion

We proposed a new active learning method in the trigonometric polynomial space. The proposed method provides exactly the optimal generalization capability and

Table 1: Computational complexity and memory required for projection learning.

| Calculation methods | Computational Complexity and Memory |
|---|---|
| Eq.(18) | $O(M^2)$ |
| Theorem 4 | $O(M)$ |
| Theorem 5[§] | $O(\mu)$ |

[§]$M = k\mu$ where $\mu$ is the dimension of $H$ and $k$ is a positive integer.

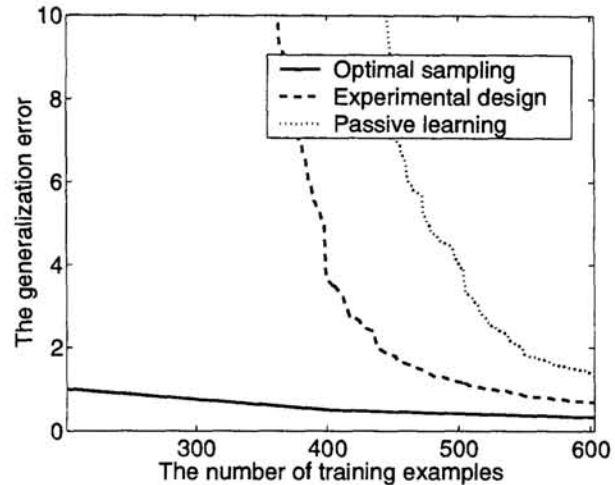

Figure 3: Relation between the number of training examples and the generalization error.

at the same time, it reduces the computational complexity and memory required for the calculation of learning results. The mechanism of achieving the optimal generalization was clarified from the functional analytic point of view.

## Footnotes

*http://ogawa-www.cs.titech.ac.jp/~sugi.

[1]For any fixed $g$ in a Hilbert space $H_1$ and any fixed $f$ in a Hilbert space $H_2$, the Neumann-Schatten product $(f \otimes \overline{g})$ is an operator from $H_1$ to $H_2$ defined by using any $h \in H_1$ as $(f \otimes \overline{g})h = \langle h, g \rangle f$.

[2]An operator $X$ is called the Moore-Penrose generalized inverse of an operator $A$ if $X$ satisfies $AXA = A$, $XAX = X$, $(AX)^* = AX$, and $(XA)^* = XA$.

# References

[1] N. Aronszajn. Theory of reproducing kernels. *Transactions on American Mathematical Society*, 68:337–404, 1950.

[2] D. Cohn. Neural network exploration using optimal experiment design. In J. Cowan *et al.* (Eds.), *Advances in Neural Information Processing Systems 6*, pp. 679–686. Morgan-Kaufmann Publishers Inc., San Mateo, CA, 1994.

[3] V. V. Fedorov. *Theory of Optimal Experiments*. Academic Press, New York, 1972.

[4] K. Fukumizu. Active learning in multilayer perceptrons. In D. Touretzky *et al.* (Eds.), *Advances in Neural Information Processing Systems 8*, pp. 295–301. The MIT Press, Cambridge, 1996.

[5] D. MacKay. Information-based objective functions for active data selection. *Neural Computation*, 4(4):590–604, 1992.

[6] H. Ogawa. Projection filter regularization of ill-conditioned problem. In *Proceedings of SPIE, 808, Inverse Problems in Optics*, pp. 189–196, 1987.

[7] H. Ogawa. Neural network learning, generalization and over-learning. In *Proceedings of the ICIIPS'92, International Conference on Intelligent Information Processing & System*, vol. 2, pp. 1–6, Beijing, China, 1992.

[8] H. Ogawa. Theory of pseudo biorthogonal bases and its application. In *Research Institute for Mathematical Science, RIMS Kokyuroku, 1067, Reproducing Kernels and their Applications*, pp. 24–38, 1998.

[9] M. Sugiyama and H. Ogawa. Functional analytic approach to model selection—Subspace information criterion. In *Proceedings of 1999 Workshop on Information-Based Induction Sciences (IBIS'99)*, pp. 93–98, Syuzenji, Shizuoka, Japan, 1999 (Its complete version is available at ftp://ftp.cs.titech.ac.jp/pub/TR/99/TR99-0009.ps.gz).

[10] M. Sugiyama and H. Ogawa. Incremental active learning in consideration of bias, *Technical Report of IEICE*, NC99-56, pp. 15–22, 1999 (Its complete version is available at ftp://ftp.cs.titech.ac.jp/pub/TR/99/TR99-0010.ps.gz).
